# Learning to Rank with Nonsmooth Cost Functions

**Christopher J.C. Burges**
Microsoft Research
One Microsoft Way
Redmond, WA 98052, USA
cburges@microsoft.com

**Robert Ragno**
Microsoft Research
One Microsoft Way
Redmond, WA 98052, USA
rragno@microsoft.com

**Quoc Viet Le**
Statistical Machine
Learning Program
NICTA, ACT 2601, Australia
quoc.le@anu.edu.au

## Abstract

The quality measures used in information retrieval are particularly difficult to optimize directly, since they depend on the model scores only through the sorted order of the documents returned for a given query. Thus, the derivatives of the cost with respect to the model parameters are either zero, or are undefined. In this paper, we propose a class of simple, flexible algorithms, called LambdaRank, which avoids these difficulties by working with implicit cost functions. We describe LambdaRank using neural network models, although the idea applies to any differentiable function class. We give necessary and sufficient conditions for the resulting implicit cost function to be convex, and we show that the general method has a simple mechanical interpretation. We demonstrate significantly improved accuracy, over a state-of-the-art ranking algorithm, on several datasets. We also show that LambdaRank provides a method for significantly speeding up the training phase of that ranking algorithm. Although this paper is directed towards ranking, the proposed method can be extended to any non-smooth and multivariate cost functions.

## 1 Introduction

In many inference tasks, the cost function[1] used to assess the final quality of the system is not the one used during training. For example for classification tasks, an error rate for a binary SVM classifier might be reported, although the cost function used to train the SVM only very loosely models the number of errors on the training set, and similarly neural net training uses smooth costs, such as MSE or cross entropy. Thus often in machine learning tasks, there are actually two cost functions: the desired cost, and the one used in the optimization process. For brevity we will call the former the 'target' cost, and the latter the 'optimization' cost. The optimization cost plays two roles: it is chosen to make the optimization task tractable (smooth, convex etc.), and it should approximate the desired cost well. This mismatch between target and optimization costs is not limited to classification tasks, and is particularly acute for information retrieval. For example, [10] list nine target quality measures that are commonly used in information retrieval, all of which depend only on the sorted order of the documents[2] and their labeled relevance. The target costs are usually averaged over a large number of queries to arrive at a single cost that can be used to assess the algorithm. These target costs present severe challenges to machine learning: they are either flat (have zero gradient with respect to the model scores), or are discontinuous, everywhere. It is very likely that a significant mismatch between the target and optimizations costs will have a substantial adverse impact on the accuracy of the algorithm.

In this paper, we propose one method for attacking this problem. Perhaps the first approach that comes to mind would be to design smoothed versions of the cost function, but the inherent 'sort' makes this very challenging. Our method bypasses the problems introduced by the sort, by defining a virtual gradient on each item *after* the sort. The method is simple and very general: it can be used for any target cost function. However, in this paper we restrict ourselves to the information retrieval domain. We show that the method gives significant benefits (for both training speed, and accuracy) for applications of commercial interest.

Notation: for the search problem, we denote the score of the ranking function by $s_{ij}$, where $i = 1, \ldots, N_Q$ indexes the query, and $j = 1, \ldots, n_i$ indexes the documents returned for that query. The general cost function is denoted $C(\{s_{ij}\}, \{l_{ij}\})$, where the curly braces denote sets of cardinality $n_i$, and where $l_{ij}$ is the label of the $j$'th document returned for the $i$'th query, where $j$ indexes the documents sorted by score. We will drop the query index $i$ when the meaning is clear. Ranked lists are indexed from the top, which is convenient when list length varies, and to conform with the notion that high rank means closer to the top of the list, we will take "higher rank" to mean "lower rank index". Terminology: for neural networks, we will use 'fprop' and 'backprop' as abbreviations for a forward pass, and for a weight-updating backward pass, respectively. Throughout this paper we also use the term "smooth" to denote $C^1$ (i.e. with first derivatives everywhere defined).

## 2 Common Quality Measures Used in Information Retrieval

We list some commonly used quality measures for information retrieval tasks: see [10] and references therein for details. We distinguish between binary and multilevel measures: for binary measures, we assume labels in $\{0, 1\}$, with 1 meaning relevant and 0 meaning not. ***Average Precision*** is a binary measure where for each relevant document, the precision is computed at its position in the ordered list, and these precisions are then averaged over all relevant documents. The corresponding quantity averaged over queries is called 'Mean Average Precision'. ***Mean Reciprocal Rank*** (MRR) is also a binary measure: if $r_i$ is the rank of the highest ranking relevant document for the $i$'th query, then the MRR is just the reciprocal rank, averaged over queries: MRR $= \frac{1}{N_Q} \sum_{i=1}^{N_Q} 1/r_i$. MRR was used, for example, in TREC evaluations of Question Answering systems, before 2002 [14]. ***Winner Takes All*** (WTA) is a binary measure for which, if the top ranked document for a given query is relevant, the WTA cost is zero, otherwise it is one. WTA is used, for example, in TREC evaluations of Question Answering systems, after 2002 [14]. ***Pair-wise Correct*** is a multilevel measure that counts the number of pairs that are in the correct order, as a fraction of the maximum possible number of such pairs, for a given query. In fact for binary classification tasks, the pair-wise correct is the same as the AUC, which has led to work exploring optimizing the AUC using ranking algorithms [15, 3]. ***bpref*** biases the pairwise correct to the top part of the ranking by choosing a subset of documents from which to compute the pairs [1, 10]. The ***Normalized Discounted Cumulative Gain*** (NDCG) is a cumulative, multilevel measure of ranking quality that is usually truncated at a particular rank level [6]. For a given query $Q_i$ the NDCG is computed as

$$\mathcal{N}_i \equiv N_i \sum_{j=1}^{L} (2^{r(j)} - 1)/\log(1 + j) \tag{1}$$

where $r(j)$ is the relevance level of the $j$'th document, and where the normalization constant $N_i$ is chosen so that a perfect ordering would result in $\mathcal{N}_i = 1$. Here $L$ is the ranking truncation level at which the NDCG is computed. The $\mathcal{N}_i$ are then averaged over the query set. NDCG is particularly well suited to Web search applications because it is multilevel and because the truncation level can be chosen to reflect how many documents are shown to the user. For this reason we will use the NDCG measure in this paper.

## 3 Previous Work

The ranking task is the task of finding a sort on a set, and as such is related to the task of learning structured outputs. Our approach is very different, however, from recent work on structured outputs, such as the large margin methods of [12, 13]. There, structures are also mapped to the reals (through choice of a suitable inner product), but the best output is found by estimating the argmax over all

possible outputs. The ranking problem also maps outputs (documents) to the reals, but solves a much simpler problem in that the number of documents to be sorted is tractable. Our focus is on a very different aspect of the problem, namely, finding ways to directly optimize the cost that the user ultimately cares about. As in [7], we handle cost functions that are multivariate, in the sense that the number of documents returned for a given query can itself vary, but the key challenge we address in this paper is how to work with costs that are everywhere either flat or non-differentiable. However, we emphasize that the method also handles the case of multivariate costs that cannot be represented as a sum of terms, each depending on the output for a single feature vector and its label. We call such functions *irreducible* (such costs are also considered by [7]). Most cost functions used in machine learning are instead reducible (for example, MSE, cross entropy, log likelihood, and the costs commonly used in kernel methods). The ranking problem itself has attracted increasing attention recently (see for example [4, 2, 8]), and in this paper we will use the RankNet algorithm of [2] as a baseline, since it is both easy to implement and performs well on large retrieval tasks.

## 4 LambdaRank

One approach to working with a nonsmooth target cost function would be to search for an optimization function which is a good approximation to the target cost, but which is also smooth. However, the *sort* required by information retrieval cost functions makes this problematic. Even if the target cost depends on only the top few ranked positions after sorting, the sort itself depends on all documents returned for the query, and that set can be very large; and since the target costs depend on only the rank order and the labels, the target cost functions are either flat or discontinuous in the scores of all the returned documents. We therefore consider a different approach. We illustrate the idea with an example which also demonstrates the perils introduced by a target / optimization cost mismatch. Let the target cost be WTA and let the chosen optimization cost be a smooth approximation to pairwise error. Suppose that a ranking algorithm $\mathcal{A}$ is being trained, and that at some iteration, for a query for which there are only two relevant documents $D_1$ and $D_2$, $\mathcal{A}$ gives $D_1$ rank one and $D_2$ rank $n$. Then on this query, $\mathcal{A}$ has WTA cost zero, but a pairwise error cost of $n - 2$. If the parameters of $\mathcal{A}$ are adjusted so that $D_1$ has rank two, and $D_2$ rank three, then the WTA error is now maximized, but the number of pairwise errors has been reduced by $n - 4$. Now suppose that at the next iteration, $D_1$ is at rank two, and $D_2$ at rank $n \gg 1$. The change in $D_1$'s score that is required to move it to top position is clearly less (possibly much less) than the change in $D_2$'s score required to move *it* to top position. Roughly speaking, we would prefer $\mathcal{A}$ to spend a little capacity moving $D_1$ up by one position, than have it spend a lot of capacity moving $D_2$ up by $n - 1$ positions. If $j_1$ and $j_2$ are the rank indices of $D_1$, $D_2$ respectively, then instead of pairwise error, we would prefer an optimization cost $C$ that has the property that

$$|\frac{\partial C}{\partial s_{j_1}}| \gg |\frac{\partial C}{\partial s_{j_2}}| \tag{2}$$

whenever $j_2 \gg j_1$. This illustrates the two key intuitions behind LambdaRank: first, it is usually much easier to specify rules determining how we would like the rank order of documents to change, after sorting them by score for a given query, than to construct a general, smooth optimization cost that has the desired properties for all orderings. By only having to specify rules for a given ordering, we are defining the gradients of an implicit cost function $C$ only at the particular points in which we are interested. Second, the rules can encode our intuition of the limited capacity of the learning algorithm, as illustrated by Eq. (2). Let us write the gradient of $C$ with respect to the score of the document at rank position $j$, for the $i$'th query, as

$$\frac{\partial C}{\partial s_j} = -\lambda_j(s_1, l_1, \cdots, s_{n_i}, l_{n_i}) \tag{3}$$

The sign is chosen so that positive $\lambda_j$ means that the document must move up the ranked list to reduce the cost. Thus, in this framework choosing an implicit cost function amounts to choosing suitable $\lambda_j$, which themselves are specified by rules that can depend on the ranked order (and scores) of *all* the documents. We will call these choices the $\lambda$ functions. At this point two questions naturally arise: first, given a choice for the $\lambda$ functions, when does there exist a function $C$ for which Eq. (3) holds; and second, given that it exists, when is $C$ convex? We have the following result from multilinear algebra (see e.g. [11]):

**Theorem** *(Poincaré Lemma): If $S \subset \mathcal{R}^n$ is an open set that is star-shaped with respect to the origin, then every closed form on $S$ is exact.*

Note that since every exact form is closed, it follows that on an open set that is star-shaped with respect to the origin, a form is closed if and only if it is exact. Now for a given query $Q_i$ and corresponding set of returned $D_{ij}$, the $n_i$ $\lambda$'s are functions of the scores $s_{ij}$, parameterized by the (fixed) labels $l_{ij}$. Let $dx^j$ be a basis of 1-forms on $\mathcal{R}^n$ and define the 1-form

$$\boldsymbol{\lambda} \equiv \sum_j \lambda_j dx^j \tag{4}$$

Then assuming that the scores are defined over $\mathcal{R}^n$, the conditions for the theorem are satisfied and $\boldsymbol{\lambda} = dC$ for some function $C$ if and only if $d\boldsymbol{\lambda} = 0$ everywhere. Using classical notation, this amounts to requiring that

$$\frac{\partial \lambda_j}{\partial s_k} = \frac{\partial \lambda_k}{\partial s_j} \quad \forall j, k \in \{1, \dots, n_i\} \tag{5}$$

This provides a simple test on the $\lambda$'s to determine if there exists a cost function for which they are the derivatives: the Jacobian (that is, the matrix $J_{jk} \equiv \partial \lambda_j / \partial s_k$) must be symmetric. Furthermore, given that such a cost function $C$ does exist, then since its Hessian is just the above Jacobian, the condition that $C$ be convex is that the Jacobian be positive semidefinite everywhere. Under these constraints, the Jacobian looks rather like a kernel matrix, except that while an entry of a kernel matrix depends on two elements of a vector space, an entry of the Jacobian can depend on all of the scores $s_j$. Note that for constant $\lambda$'s, the above two conditions are trivially satisfied, and that for other choices that give rise to symmetric $J$, positive definiteness can be imposed by adding diagonal regularization terms of the form $\lambda_j \mapsto \lambda_j + \alpha_j s_j$, $\alpha_j > 0$.

LambdaRank has a clear physical analogy. Think of the documents returned for a given query as point masses. $\lambda_j$ then corresponds to a force on the point mass $D_j$. If the conditions of Eq. (5) are met, then the forces in the model are conservative, that is, they may be viewed as arising from a potential energy function, which in our case is the implicit cost function $C$. For example, if the $\lambda$'s are linear in the outputs $s$, then this corresponds to a spring model, with springs that are either compressed or extended. The requirement that the Jacobian is positive semidefinite amounts to the requirement that the system of springs have a unique global minimum of the potential energy, which can be found from any initial conditions by gradient descent (this is not true in general, for arbitrary systems of springs). The physical analogy provides useful guidance in choosing $\lambda$ functions. For example, for a given query, the forces ($\lambda$'s) should sum to zero, since otherwise the overall system (mean score) will accelerate either up or down. Similarly if a contribution to a document $A$'s $\lambda$ is computed based on its position with respect to document $B$, then $B$'s $\lambda$ should be incremented by an equal and opposite amount, to prevent the pair itself from accelerating (Newton's third law, [9]).

Finally, we emphasize that LambdaRank is a very simple method. It requires only that one provide rules for the derivatives of the implicit cost for any given sorted order of the documents, and as we will show, such rules are easy to come up with.

## 5 A Speedup for RankNet Learning

RankNet [2] uses a neural net as its function class. Feature vectors are computed for each query/document pair. RankNet is trained on those pairs of feature vectors, for a given query, for which the corresponding documents have different labels. At runtime, single feature vectors are fpropped through the net, and the documents are ordered by the resulting scores. The RankNet cost consists of a sigmoid (to map the outputs to $[0, 1]$) followed by a pair-based cross entropy cost, and takes the form given in Eq. (8) below. Training times for RankNet thus scale quadratically with the mean number of pairs per query, and linearly with the number of queries.

The ideas proposed in Section 4 suggest a simple method for significantly speeding up RankNet training, making it also approximately linear in the number of labeled documents per query, rather than in the number of pairs per query. This is a very significant benefit for large training sets. In fact the method works for any ranking method that uses gradient descent and for which the cost depends on pairs of items for each query. Most neural net training, RankNet included, uses a stochastic gradient update, which is known to give faster convergence. However here we will use batch learning

per query (that is, the weights are updated for each query). We present the idea for a general ranking function $f : \mathcal{R}^n \mapsto \mathcal{R}$ with optimization cost $C : \mathcal{R} \mapsto \mathcal{R}$. It is important to note that adopting batch training alone does not give a speedup: to compute the cost and its gradients we would still need to fprop each pair. Consider a single query for which $n$ documents have been returned. Let the output scores of the ranker be $s_j$, $j = 1, \ldots, n$, the model parameters be $w_k \in \mathcal{R}$, and let the set of pairs of document indices used for training be $\mathcal{P}$. The total cost is $C_T \equiv \sum_{\{i,j\} \in P} C(s_i, s_j)$ and its derivative with respect to $w_k$ is

$$\frac{\partial C_T}{\partial w_k} = \sum_{\{i,j\} \in P} \frac{\partial C(s_i, s_j)}{\partial s_i} \frac{\partial s_i}{\partial w_k} + \frac{\partial C(s_i, s_j)}{\partial s_j} \frac{\partial s_j}{\partial w_k} \tag{6}$$

It is convenient to refactor the sum: let $\mathcal{P}_i$ be the set of indices $j$ for which $\{i, j\}$ is a valid pair, and let $\mathcal{D}$ be the set of document indices. Then we can write the first term as

$$\frac{\partial C_T}{\partial w_k} = \sum_{i \in \mathcal{D}} \frac{\partial s_i}{\partial w_k} \sum_{j \in \mathcal{P}_i} \frac{\partial C(s_i, s_j)}{\partial s_i} \tag{7}$$

and similarly for the second. The algorithm is as follows: instead of backpropping each pair, first $n$ fprops are performed to compute the $s_i$ (and for the general LambdaRank algorithm, this would also be where the sort on the scores is performed); then for each $i = 1, \ldots, n$ the $\lambda_i \equiv \sum_{j \in P_i} \frac{\partial C(s_i, s_j)}{\partial s_i}$ are computed; then to compute the gradients $\frac{\partial s_i}{\partial w_k}$, $n$ fprops are performed, and finally the $n$ back-props are done. The key point is that although the overall computation still has an $n^2$ dependence arising from the second sum in (7), computing the terms $\frac{\partial C(s_i, s_j)}{\partial s_i} = \frac{-1}{1 + e^{s_1 - s_2}}$ is far cheaper than the computation required to perform the $2n$ fprops and $n$ backprops. Thus we have effectively replaced a $O(n^2)$ algorithm with an $O(n)$ one[3].

## 6 Experiments

We performed experiments to (1) demonstrate the training speedup for RankNet, and (2) assess whether LambdaRank improves the NDCG test performance. For the latter, we used RankNet as a baseline. Even though the RankNet optimization cost is not NDCG, RankNet is still very effective at optimizing NDCG, using the method proposed in [2]: after each epoch, compute the NDCG on a validation set, and after training, choose the net for which the validation NDCG is highest. Rather than attempt to derive from first principles the optimal Lambda function for the NDCG target cost (and for a given dataset), which is beyond the scope of this paper, we wrote several plausible $\lambda$-functions and tested them on the Web search data. We then picked the single $\lambda$ function that gave the best results on that particular validation set, and then used that $\lambda$ function for all of our experiments; this is described below.

### 6.1 RankNet Speedup Results

Here the training scheme is exactly LambdaRank training, but with the RankNet gradients, and with no sort: we call the corresponding $\lambda$ function $\mathcal{G}$. We will refer to the original RankNet training as V1 and LambdaRank speedup as V2. We compared V1 and V2 in two sets of experiments. In the first we used 1000 queries taken from the Web data described below, and in the second we varied the number of documents for a given query, using the artificial data described below. Experiments were run on a 2.2GHz 32 bit Opteron machine. We compared V1 to V2 for 1 layer and 2 layer (with 10 hidden nodes) nets. V1 was also run *using batch update per query*, to clearly show the gain (the convergence as a function of epoch was found to be similar for batch and non-batch updates; furthermore running time for batch and non-batch is almost identical). For the single layer net, on the Web data, LambdaRank with $\mathcal{G}$ was measured to be 5.1 times faster, and for two layer, 8.0 times faster: the left panel of Figure 1 shows the results (where max validation NDCG is plotted). Each point on the graph is one epoch. Results for the two layer nets were similar. The right panel shows a log log plot of training time versus number of documents, as the number of documents per query

varies from 4,000 to 512,000 in the artificial set. Fitting the curves using linear regression gives the slopes of V1 and V2 to be 1.943 and 1.185 respectively. Thus V1 is close to quadratic (but not exactly, due to the fact that only a subset of pairs is used, namely, those with documents whose labels differ), and V1 is close to linear, as expected.

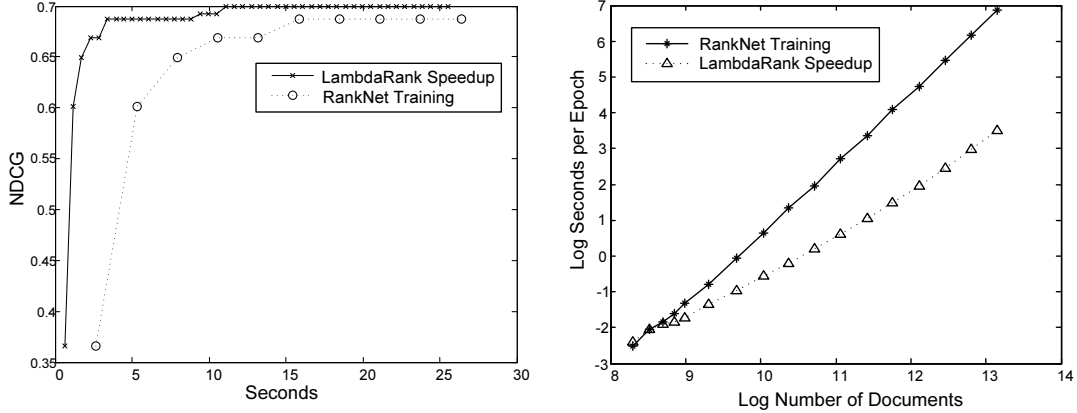

Figure 1: Speeding up RankNet training. Left: linear nets. Right: two layer nets.

## 6.2  $\lambda$-function Chosen for Ranking Experiments

To implement LambdaRank training, we must first choose the $\lambda$ function (Eq. (3)), and then substitute in Eq. (5). Using the physical analogy, specifying a $\lambda$ function amounts to specifying rules for the 'force' on a document given its neighbors in the ranked list. We tried two kinds of $\lambda$ function: those where a document's $\lambda$ gets a contribution from all pairs with different labels (for a given query), and those where its $\lambda$ depends only on its nearest neighbors in the sorted list. All $\lambda$ functions were designed with the NDCG cost function in mind, and most had a margin built in (that is, a force is exerted between two documents even if they are in the correct order, until their difference in scores exceeds that margin). We investigated step potentials, where the step sizes are proportional to the NDCG gain found by swapping the pair; spring models; models that estimated the NDCG gradient using finite differences; and models where the cost was estimated as the gradient of a smooth, pairwise cost, also scaled by NDCG gain from swapping the two documents. We tried ten different $\lambda$ functions in all. Due to space limitations we will not give results on all these functions here: instead we will use the one that worked best on the Web validation data for all experiments. This function used the RankNet cost, scaled by the NDCG gain found by swapping the two documents in question. The RankNet cost combines a sigmoid output and the cross entropy cost, and is similar to the negative binomial log-likelihood cost [5], except that it is based on pairs of items: if document $i$ is to be ranked higher than document $j$, then the RankNet cost is [2]:

$$C_{i,j}^R = s_j - s_i + \log(1 + e^{s_i - s_j}) \qquad (8)$$

and if the corresponding document ranks are $r_i$ and $r_j$, then taking derivatives of Eq. (8) and combining with Eq. (1) gives

$$\lambda = N \left( \frac{1}{1 + e^{s_i - s_j}} \right) \left( 2^{l_i} - 2^{l_j} \right) \left( \frac{1}{\log(1 + i)} - \frac{1}{\log(1 + j)} \right) \qquad (9)$$

where $N$ is the reciprocal max DCG for the query. Thus for each pair, *after* the sort, we increment each document's force by $\pm\lambda$, where the more relevant document gets the positive increment.

## 6.3  Ranking for Search Experiments

We performed experiments on three datasets: artificial, web search, and intranet search data. The data are labeled from 0 to $M$, in order of increasing relevance: the Web search and artificial data have $M = 4$, and the intranet search data, $M = 3$. The corresponding NDCG gains (the numerators in Eq. (1)) were therefore 0, 3, 7, 15 and 31. In all graphs, 95% confidence intervals are shown. In all experiments, we varied the learning rate from as low as 1e-7 to as high as 1e-2, and for each

experiment we picked that rate that gave the best validation results. For all training, the learning rate was reduced be a factor of 0.8 if the training cost (Eq. (8), for RankNet, and the NDCG at truncation level 10, for LambdaRank) increased over the value for the previous epoch. Training was done for 300 epochs for the artificial and Web search data, and for 200 epochs for the intranet data, and training was restarted (with random weights) if the cost did not reduce for 50 iterations.

### 6.3.1 Artificial Data

We used artificial data to remove any variance stemming from the quality of the features or of the labeling. We followed the prescription given in [2] for generating random cubic polynomial data. However, here we use five levels of relevance instead of six, a label distribution corresponding to real datasets, and more data, all to more realistically approximate a Web search application. We used 50 dimensional data, 50 documents per query, and 10K/5K/10K queries for train/valid/test respectively. We report the NDCG results in Figure 2 for ten NDCG truncation levels. In this clean dataset, LambdaRank clearly outperforms RankNet. Note that the gap increases at higher relevance levels, as one might expect due to the more direct optimization of NDCG.

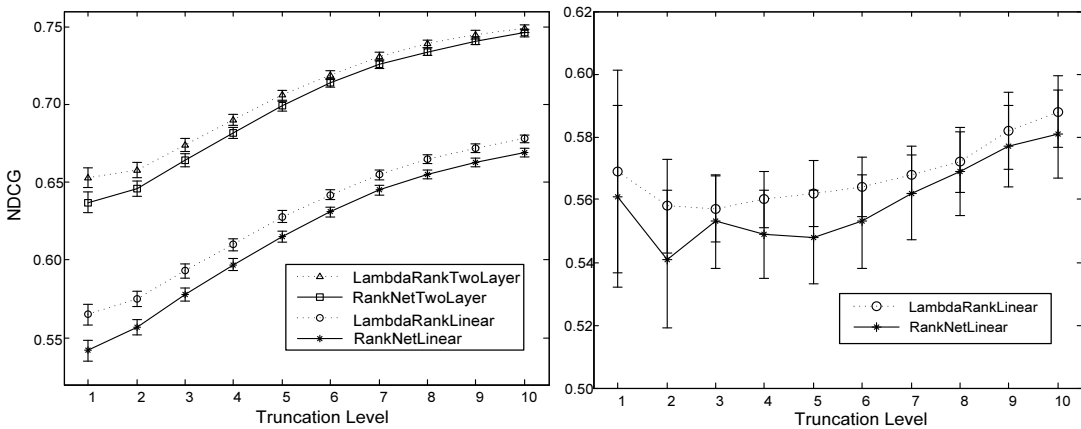

Figure 2: Left: Cubic polynomial data. Right: Intranet search data.

### 6.3.2 Intranet Search Data

This data has dimension 87, and only 400 queries in all were available. The average number of documents per query is 59.4. We used 5 fold cross validation, with 2+2+1 splits between train/validation/test sets. We found that it was important for such a small dataset to use a relatively large validation set to reduce variance. The results for the linear nets are shown in Figure 2: although LambdaRank gave uniformly better mean NDCGs, the overlapping error bars indicate that on this set, LambdaRank does not give statistically significantly better results than RankNet at 95% confidence. For the two layer nets the NDCG means are even closer. This is an example of a case where larger datasets are needed to see the difference between two algorithms (although it's possible that more powerful statistical tests would find a difference here also).

### 6.4 Web Search Data

This data is from a commercial search engine and has 367 dimensions, with on average 26.1 documents per query. The data was created by shuffling a larger dataset and then dividing into train, validation and test sets of size 10K/5K/10K queries, respectively. In Figure 3, we report the NDCG scores on the dataset at truncation levels from 1 to 10. We show separate plots to clearly show the differences: in fact, the linear LambdaRank results lie on top of the two layer RankNet results, for the larger truncation values.

## 7 Conclusions

We have demonstrated a simple and effective method for learning non-smooth target costs. LambdaRank is a general approach: in particular, it can be used to implement RankNet training, and it

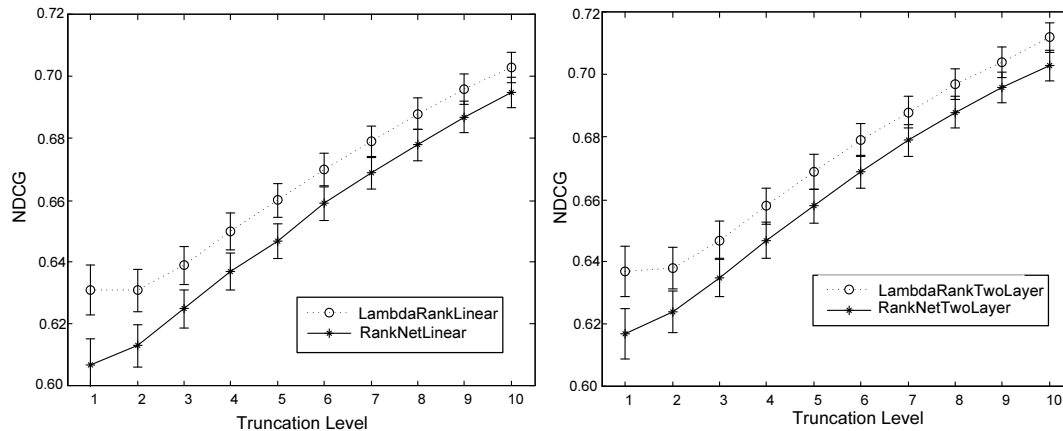

Figure 3: NDCG for RankNet and LambdaRank. Left: linear nets. Right: two layer nets

furnishes a significant training speedup there. We studied LambdaRank in the context of the NDCG target cost for neural network models, but the same ideas apply to any non-smooth target cost, and to any differentiable function class. It would be interesting to investigate using the same method starting with other classifiers such as boosted trees.

### Acknowledgments

We thank M. Taylor, J. Platt, A. Laucius, P. Simard and D. Meyerzon for useful discussions and for providing data.

## Footnotes

[1]Throughout this paper, we will use the terms "cost function" and "quality measure" interchangeably, with the understanding that the cost function is some monotonic decreasing function of the corresponding quality measure.

[2]For concreteness we will use the term 'documents' for the items returned for a given query, although the returned items can be more general (e.g. multimedia items).

[3]Two further speedups are possible, and are not explored here: first, only the first $n$ fprops need be performed if the node activations are stored, since those stored activations could then be used during the $n$ back-props; second, the $e^{s_i}$ could be precomputed before the pairwise sum is done.

## References

[1] C. Buckley and E. Voorhees. Evaluating evaluation measure stability. In *SIGIR*, pages 33–40, 2000.

[2] C.J.C. Burges, T. Shaked, E. Renshaw, A. Lazier, M. Deeds, N. Hamilton, and G. Hullender. Learning to Rank using Gradient Descent. In *ICML 22*, Bonn, Germany, 2005.

[3] C. Cortes and M. Mohri. Confidence Intervals for the Area Under the ROC Curve. In *NIPS 18*. MIT Press, 2005.

[4] Y. Freund, R. Iyer, R.E. Schapire, and Y. Singer. An efficient boosting algorithm for combining preferences. *Journal of Machine Learning Research*, 4:933–969, 2003.

[5] J. Friedman, T. Hastie, and R. Tibshirani. Additive logistic regression: A statistical view of boosting. *The Annals of Statistics*, 28(2):337–374, 2000.

[6] K. Jarvelin and J. Kekalainen. IR evaluation methods for retrieving highly relevant documents. In *SIGIR 23*. ACM, 2000.

[7] T. Joachims. A support vector method for multivariate performance measures. In *ICML 22*, 2005.

[8] I. Matveeva, C. Burges, T. Burkard, A. Lauscius, and L. Wong. High accuracy retrieval with multiple nested rankers. In *SIGIR*, 2006.

[9] I. Newton. *Philosophiae Naturalis Principia Mathematica*. The Royal Society, 1687.

[10] S. Robertson and H. Zaragoza. On rank-based effectiveness measures and optimisation. Technical Report MSR-TR-2006-61, Microsoft Research, 2006.

[11] M. Spivak. *Calculus on Manifolds*. Addison-Wesley, 1965.

[12] B. Taskar, V. Chatalbashev, D. Koller, and C. Guestrin. Learning structured prediciton models: A large margin approach. In *ICML 22*, Bonn, Germany, 2005.

[13] I. Tsochantaridis, T. Hofmann, T. Joachims, and Y. Altun. Support vector machine learning for interdependent and structured output spaces. In *ICML 24*, 2004.

[14] E.M. Voorhees. Overview of the TREC 2001/2002 Question Answering Track. In *TREC*, 2001,2002.

[15] L. Yan, R. Dodlier, M.C. Mozer, and R. Wolniewicz. Optimizing Classifier Performance via an Approximation to the Wilcoxon-Mann-Whitney Statistic. In *ICML 20*, 2003.
